# A spatially varying two-sample recombinant coalescent, with applications to HIV escape response

**Alexander Braunstein**
Statistics Department
University of Pennsylvania
Wharton School
Philadelphia, PA 19104
braunsf@wharton.upenn.edu

**Zhi Wei**
Computer Science Department
New Jersey Institute of Technology
Newark, NJ 07102
zhiwei@njit.edu

**Shane T. Jensen**
Statistics Department
University of Pennsylvania
Wharton School
Philadelphia, PA 19104
stjensen@wharton.upenn.edu

**Jon D. McAuliffe**
Statistics Department
University of Pennsylvania
Wharton School
Philadelphia, PA 19104
mcjon@wharton.upenn.edu

## Abstract

Statistical evolutionary models provide an important mechanism for describing and understanding the escape response of a viral population under a particular therapy. We present a new hierarchical model that incorporates spatially varying mutation and recombination rates at the nucleotide level. It also maintains separate parameters for treatment and control groups, which allows us to estimate treatment effects explicitly. We use the model to investigate the sequence evolution of HIV populations exposed to a recently developed antisense gene therapy, as well as a more conventional drug therapy. The detection of biologically relevant and plausible signals in both therapy studies demonstrates the effectiveness of the method.

## 1   Introduction

The human immunodeficiency virus (HIV) has one of the highest levels of genetic variability yet observed in nature. This variability stems from its unusual population dynamics: a high growth rate ($\sim$10 billion new viral particles, or virions, per patient per day) combined with a replication cycle that involves frequent nucleotide mutations as well as recombination between different HIV genomes that have infected the same cell.

The rapid evolution of HIV and other viruses gives rise to a so-called escape response when infected cells are subjected to therapy. Widespread availability of genome sequencing technology has had a profound effect on the study of viral escape response. Increasingly, virologists are gathering two-sample data sets of viral genome sequences: a control sample contains genomes from a set of virions gathered before therapy, and a treatment sample consists of genomes from the post-therapeutic viral population. HIV treatment samples gathered just days after the start of therapy can exhibit a significant escape response.

Up to now, statistical analyses of two-sample viral sequence data sets have been mainly rudimentary. As a representative example, [7] presents tabulated counts of mutation occurrences (relative to a reference wild-type sequence) in the control group and the treatment group, without attempting any statistical inference.

In this paper we develop a model which allows for a detailed quantification of the escape response present in a two-sample data set. The model incorporates mutation and recombination rate parameters which vary positionally along the viral genome, and which differ between the treatment and control samples. We present a reversible-jump MCMC procedure for approximate posterior inference of these parameters. The resulting posterior distribution suggests specific regions of the genome where the treatment sample's evolutionary dynamics differ from the control's: this is the putative escape response. Thus, the model permits an analysis that can point the way to improvements of current therapies and to the development of new therapeutic strategies for HIV and other viruses.

In the remainder of the paper, we first provide the details of our statistical model and inference procedure. Then we illustrate the use of the model in two applications. The first study consists of a control sample of viral sequences obtained from HIV-infected individuals before a drug treatment, and a corresponding post-treatment sample [9]. The second study set is an *in vitro* investigation of a new gene therapy for HIV; it contains a control sample of untreated virions and a treatment sample of virions challenged with the therapy [7].

## 2  Methods

We begin by briefly describing the standard statistical genetics framework for populations evolving under mutation and recombination. Then we present a new Bayesian hierarchical model for two groups of sequences, each group sampled from one of two related populations. We derive an MCMC procedure for approximate posterior inference in the model; this procedure is implemented in the program PICOMAP. Our approach involves modifications and generalizations of the OMEGAMAP method [12], as we explain. In what follows, each "individual" in a population is a sequence of $L$ nucleotides (plus a gap symbol, used when sequences have insertions or deletions relative to each other). The positions along a sequence are called *sites*. An *alignment* is a matrix in which rows are sequences, columns are sites, and the $(i, j)$th entry is individual $i$'s nucleotide at site $j$.

### 2.1  The coalescent with recombination

The genome sequences in the control sample were drawn at random from a large population of sequences at a fixed point in time. We approximate the evolution of this population using the Wright-Fisher evolutionary model with recombination [3]. Similarly, the treatment sample sequences are viewed as randomly drawn from a Wright-Fisher recombining population, but governed by different evolutionary parameters.

In the basic Wright-Fisher model without recombination, a fixed-size population evolves in discrete, nonoverlapping generations. Each sequence in the $g$th generation is determined by randomly choosing a sequence from the $(g - 1)$th generation, mutating it at one position with probability $u$, and leaving it unchanged with probability $1 - u$. Typically, many individuals in each generation share a parent from the previous generation.

A key insight in statistical population genetics, due to Kingman [5], is the following. If we have a small sample from a large Wright-Fisher population at a fixed time, and we want to do calculations involving the probability distribution over the sample's unknown ancestral history, it is highly uneconomical to "work forwards" from older generations – most individuals will not be part of the sample's genealogy. Instead, we should follow the lineages of the sampled individuals backwards in time as they repeatedly coalesce at common ancestors, forming a tree rooted at the most recent common ancestor (MRCA) of the sample. Kingman showed that the continuous-time limit of the Wright-Fisher model induces a simple distribution, called the *coalescent process*, on the topology and branch lengths of the resulting tree. Mutation events in the coalescent can be viewed as a separate point process marking locations on the branches of a given coalescent tree. This point process is independent of the tree-generating coalescent process.

Recombination, however, substantially complicates matters. The Wright-Fisher dynamics are extended to model recombination as follows. Choose one "paternal" and one "maternal" sequence from generation $(g - 1)$. With probability $r$, their child sequence in generation $g$ is a recombinant: a juncture between two adjacent sites is chosen uniformly at random, and the child is formed by joining the paternal sequence to the left of the juncture with the maternal sequence to the right. With probability $(1 - r)$, the child is a copy of just one of the two parents, possibly mutated as above.

Now look backwards in time at the ancestors of a sample: we find both coalescence events, where two sequences merge into a common ancestor, and recombination events, where a single sequence splits into the two parent sequences that formed it. Thus the genealogy is not a tree but a graph, the *ancestral recombination graph* (ARG). The continuous-time limit of the Wright-Fisher model with recombination induces a distribution over ARGs called the *recombinant coalescent* [4, 2].

In fact, the ARG is the union of $L$ coalescent trees. A single site is never split by recombination, so we can follow that site in the sample backwards in time through coalescence events to its MRCA. But recombination causes the sample to have a possibly different ancestral tree (and different MRCA) at each site. The higher the rate of recombination (corresponding to the parameter $r$), the more often the tree changes along the alignment. For this reason, methods that estimate a fixed, global phylogeny are badly biased in samples from highly recombinant populations, like viruses [10].

The Wright-Fisher assumptions appear quite stylized. But experience has shown that the coalescent and the recombinant coalescent can give reasonable results when applied to samples from populations not matching the Wright-Fisher model, such as populations of increasing size [3].

## 2.2 A two-sample hierarchical recombinant coalescent

We now present the components of our new hierarchical model for a control sample and a treatment sample of nucleotide sequences drawn from two recombining populations. To our knowledge, this is the first fully specified probabilistic model for such data. There are four parameter vectors of primary interest in the model: a control-population mutation rate $\mu^C$ which varies along the sequence, a corresponding spatially varying treatment-population vector $\mu^T$, and analogous recombination rate parameter vectors $\rho^C$ and $\rho^T$. (The $\mu$ and $\rho$ here correspond to the $u$ and $r$ mentioned above.)

The prior distribution on $\mu^C$ and $\mu^T$ takes the following hierarchical form:

$$(B_\mu, S_\mu) \mid q_\mu \qquad \sim \text{Blocks}(q_\mu), \tag{1}$$

$$\log \mu_i \mid \mu_0, \sigma^2_{\mu_0} \sim N(\log \mu_0, \sigma^2_{\mu_0}), \qquad i = 1, \ldots, B_\mu, \tag{2}$$

$$(\log \mu_i^C, \log \mu_i^T) \mid \mu_i, \sigma^2_\mu \stackrel{\text{iid}}{\sim} N(\log \mu_i, \sigma^2_\mu), \qquad i = i, \ldots, B_\mu. \tag{3}$$

This prior is designed to give $\mu^C$ and $\mu^T$ a block structure: the Blocks distribution divides the $L$ sequence positions into $B_\mu$ adjacent subsequences, with the index of each subsequence's rightmost site given by $S_\mu = (S_\mu^1, \ldots, S_\mu^{B_\mu})$, $1 \leq S_\mu^1 < \cdots \leq S_\mu^{B_\mu} \leq L$. Under the Blocks distribution, $(B_\mu - 1)$ is a $\text{Bin}(L-1, q_\mu)$ random variable, and given $B_\mu$, the indexes $S_\mu$ are a simple random sample without replacement from $\{1, \ldots, L\}$. The sites in the $i$th block all mutate at the same rate $\mu_i^C$ (in the control population) or $\mu_i^T$ (in the treatment population). We lose no generality in sharing the same block structure between the populations: two separate block structures can be replaced with a single block structure formed from the union of their $S_\mu$'s. To generate the per-population mutation rates within a block, we first draw a lognormally distributed variable $\mu_i$, which then furnishes the mean for the independent lognormal variables $\mu_i^C$ and $\mu_i^T$. The triples $(\mu_i, \mu_i^C, \mu_i^T)$ are mutually independent across blocks $i = 1, \ldots, B_\mu$.

The recombination rate parameters $(\rho^C, \rho^T)$ are independent of $(\mu^C, \mu^T)$ and have the same form of prior distribution (1)–(3), mutatis mutandis. In our empirical analyses, we set the hyperparameters $q_\mu$ and $q_\rho$ to get prior means of 20 to 50 blocks; results were not sensitive to these settings. We put simple parametric distributions on the hyperparameters $\mu_0, \sigma^2_{\mu_0}, \sigma^2_\mu$, and their $\rho$ analogs, and included them in the sampling procedure.

The remaining component of the model is the likelihood of the two observed samples. Let $H^C$ be the alignment of control-sample sequences and $H^T$ the treatment-sample sequence alignment. Conditional on all parameters, $H^C$ and $H^T$ are independent. Focus for a moment on $H^C$. Since we wish to view it as a sample from a Wright-Fisher recombining population, its likelihood corresponds to the probability, under the coalescent-with-recombination distribution, of the set of all ARGs that could have generated $H^C$. However, using the nucleotide mutation model described below, even Monte Carlo approximation of this probability is computationally intractable [12].

So instead we approximate the true likelihood with a distribution called the "product of approximate conditionals," or PAC [6]. PAC orders the $K$ sequences in $H^C$ arbitrarily, then approximates their probability as the product of probabilities from $K$ hidden Markov models. The $k$th HMM evaluates

the probability that sequence $k$ was produced by mutating and recombining sequences 1 through $k-1$. We thus obtain the final components of our hierarchical model:

$$H^{\mathrm{C}} \mid \mu^{\mathrm{C}}, \rho^{\mathrm{C}}, \eta \ \sim \ \mathrm{PAC}(\mu^{\mathrm{C}}, \rho^{\mathrm{C}}, \eta) \,, \tag{4}$$

$$H^{\mathrm{T}} \mid \mu^{\mathrm{T}}, \rho^{\mathrm{T}}, \eta \ \sim \ \mathrm{PAC}(\mu^{\mathrm{T}}, \rho^{\mathrm{T}}, \eta) \,. \tag{5}$$

In order to apply PAC, we must specify a nucleotide substitution model, that is, the probability that a nucleotide $i$ mutates to a nucleotide $j$ over evolutionary distance $t$. In the above, $\eta$ parametrizes this model. For our analyses, we employed the well-known Felsenstein substitution model, augmented with a fifth symbol to represent gaps [8]. For simplicity, we constructed fixed empirical estimates of the Felsenstein parameters $\eta$, in a standard way.

To incorporate the extended Felsenstein model in PAC, it is necessary to integrate evolutionary distance out of the substitution process $p(j \mid i, 2t)$, using the exponential distribution induced by the coalescent on the evolutionary distance $2t$ between pairs of sampled individuals. It can be shown that the required quantity is

$$p(j \mid i) = \int p(j \mid i, 2t) p(t) \, dt =$$
$$\left(1 - \frac{k}{k+2\beta}\right) \pi_j + \left(\frac{k}{k+2(\alpha \mathbf{1}[i \neq \mathrm{gap}] + \beta)}\right) \mathbf{1}[i = j] +$$
$$\left(\frac{k}{k+2\beta} - \frac{k}{k+2(\alpha+\beta)}\right) \left(\frac{\pi_j}{\pi_i + \pi_j}\right) \mathbf{1}[(i,j) \in \{(A,G),(C,T)\}] \,. \tag{6}$$

Here $k$ is the number of sampled individuals, and $\pi_i$, $\pi_j$, $\alpha$, and $\beta$ are Felsenstein model parameters (the last two depending on the mutation rate at the site in question). $\mathbf{1}[\cdot]$ is the indicator function of the predicate in brackets.

The blocking prior (1) and the use of PAC with spatially varying parameters are ideas drawn from OMEGAMAP [12]. But our approach differs in two significant respects. First, OMEGAMAP models codons (the protein sequence encoded by nucleotides), not the nucleotides themselves. This is sometimes unsuitable. For example, in one of our empirical analyses, the treatment population receives RNA antisense gene therapy. The target of this therapy is the primary HIV genome sequence itself, not its protein products. So we would expect the escape response to manifest at the nucleotide level, in the targeted region of the genome. Our model can capture this. Second, we perform simultaneous hierarchical inference about the control and treatment sample, which encourages the parameter estimates to differ between the samples only where strongly justified by the data. Using a one-sample tool like OMEGAMAP on each sample in isolation would tend to increase the number of artifactual differences between corresponding parameters in each sample.

## 2.3 Inference

The posterior distribution of the parameters in our model cannot be calculated analytically. We therefore employ a reversible-jump Metropolis-within-Gibbs sampling strategy to construct an approximate posterior. In such an approach, sets of parameters are iteratively sampled from their posterior conditional distributions, given the current values of all other parameters. Because the Blocks prior generates mutation and recombination parameters with piecewise-constant profiles along the sequence, we call our sampler implementation PICOMAP.

The sampler uses Metropolis-Hastings updates for the numerical values of parameters, and reversible-jump updates [1] to explore the blocking structures $(B_\mu, S_\mu)$ and $(B_\rho, S_\rho)$. The block updates consider extending a block to the left or right, merging two adjacent blocks, and splitting a block. They are similar to the updates (B2)-(B4) of [12], so we omit the details.

To illustrate one of the parameter updates within a block, let $(\mu_i^{\mathrm{C}}, \mu_i^{\mathrm{T}})$ be the current values of the control and treatment mutation rates in block $i$. We sample proposal values

$$\log \tilde{\mu}_i^{\mathrm{C}} \sim N(\log \mu_i^{\mathrm{C}}, \tau^2) \,, \tag{7}$$

$$\log \tilde{\mu}_i^{\mathrm{T}} \sim N(\log \mu_i^{\mathrm{T}}, \tau^2) \,, \tag{8}$$

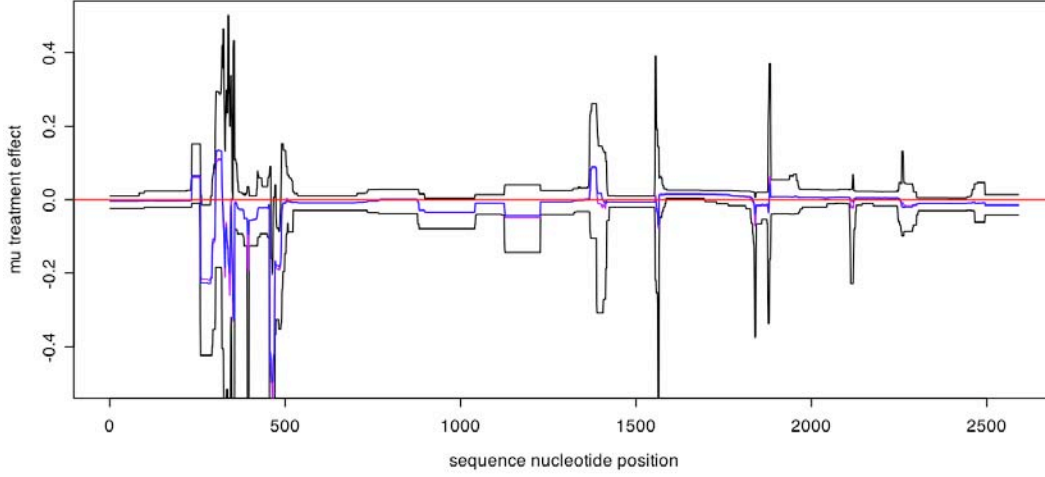

Figure 1: Posterior estimate of the effect of enfuvirtide drug therapy on mutation rates. Blue line is posterior mean, Black lines are 95% highest-posterior-density (HPD) intervals.

where $\tau^2$ is a manually configured tuning parameter for the proposal distribution. These proposals are accepted with probability

$$\frac{p(H^{\mathrm{C}} \mid \tilde{\mu}_i^{\mathrm{C}}, \theta) \, p(H^{\mathrm{T}} \mid \tilde{\mu}_i^{\mathrm{T}}, \theta)}{p(H^{\mathrm{C}} \mid \mu_i^{\mathrm{C}}, \theta) \, p(H^{\mathrm{T}} \mid \mu_i^{\mathrm{T}}, \theta)} \cdot \frac{p(\tilde{\mu}_i^{\mathrm{C}}, \tilde{\mu}_i^{\mathrm{T}} \mid \mu_i)}{p(\mu_i^{\mathrm{C}}, \mu_i^{\mathrm{T}} \mid \mu_i)}, \tag{9}$$

where

$$\frac{p(\tilde{\mu}_i^{\mathrm{C}}, \tilde{\mu}_i^{\mathrm{T}} \mid \mu_i)}{p(\mu_i^{\mathrm{C}}, \mu_i^{\mathrm{T}} \mid \mu_i)} = \frac{\mu_i^{\mathrm{T}} \, \mu_i^{\mathrm{C}}}{\tilde{\mu}_i^{\mathrm{T}} \, \tilde{\mu}_i^{\mathrm{C}}} \frac{\exp\left\{-((\log \tilde{\mu}_i^{\mathrm{T}} - \log \mu_i)^2 + (\log \tilde{\mu}_i^{\mathrm{C}} - \log \mu_i)^2)/2\sigma^2\right\}}{\exp\left\{-((\log \mu_i^{\mathrm{T}} - \log \mu_i)^2 + (\log \mu_i^{\mathrm{C}} - \log \mu_i)^2)/2\sigma^2\right\}}. \tag{10}$$

Here $\theta$ denotes the current values of all other model parameters. Notice that symmetry in the proposal distribution causes that part of the MH acceptance ratio to cancel.

The PICOMAP sampler involves a number of other update formulas, which we do not describe here due to space constraints.

# 3 Results

In this section, we apply the PICOMAP methodology to HIV sequence data from two different studies. In the first study, several HIV-infected patients were exposed to a drug-based therapy. In the second study, the HIV virus was exposed *in vitro* to a novel antisense gene therapy. In both cases, our analysis extracts biologically relevant features of the evolutionary response of HIV to these therapeutic challenges.

For each study we ran at least 8 chains to monitor convergence of the sampler. The chains converged without exception and were thinned accordingly, then combined for analysis. In the interest of brevity, we include only plots of the posterior treatment-effect estimates for both mutation and recombination rates.

## 3.1 Drug therapy study

In this study, five patients had blood samples taken both before and after treatment with the drug enfuvirtide, also known as Fuzeon or T-20 [11]. Sequences of the Envelope (Env) region of the HIV genome were generated from each of these blood samples. Pooling across these patients, we have 28 pre-exposure Env sequences which we label as the control sample, and 29 post-exposure Env sequences which we label as the treatment sample. We quantify the treatment effect of exposure

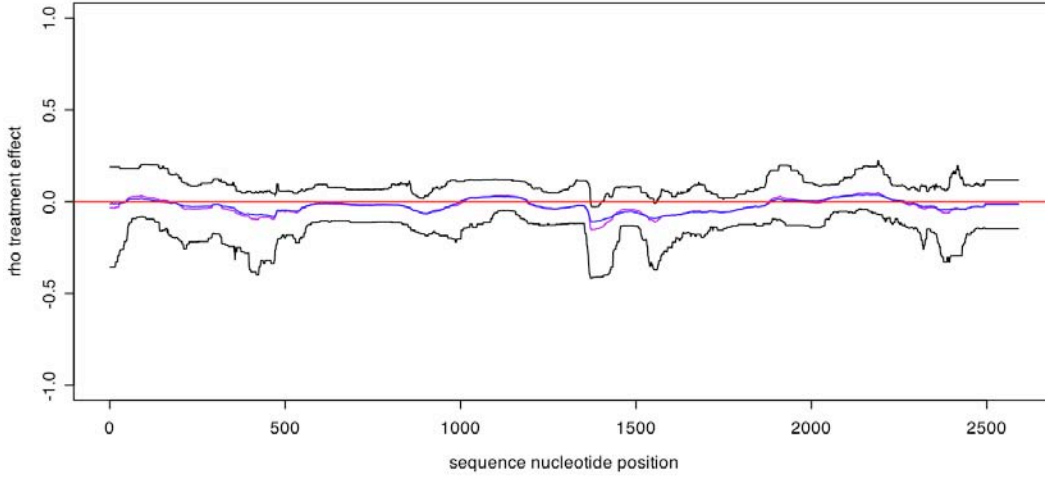

Figure 2: Posterior estimate of the effect of enfuvirtide drug therapy on recombination rates. Blue line is posterior mean, Black lines are 95% HPD intervals.

to the drug by calculating the posterior mean and 95% highest-posterior-density (HPD) intervals of the difference in recombination rates $\rho^T - \rho^C$ and mutation rates $\mu^T - \mu^C$ at each position of the genomic sequence.

The very existence in the patient of a post-exposure HIV population indicates the evolution of sequence changes that have conferred resistance to the action of the drug enfuvirtide. In fact, resistance-conferring mutations are known *a priori* to occur at nucleotide locations 1639-1668 in the Env sequence. Figure 1 shows the posterior estimate of the treatment effect on mutation rates over the length of the Env sequence. From nucleotide positions 1590-1700, the entire 95% HPD interval of the mutation rate treatment effect is above zero, which suggests our model is able to detect elevated levels of mutation in the resistance-conferring region, among individuals in the treatment sample.

Another preliminary observation from this study was that both the pre-exposure and post-exposure sequences are mixtures of several different HIV subtypes. Subtype identity is specified by the V3 loop subsequence of the Env sequence, which corresponds to nucleotide positions 887-995. Since it is unlikely that resistance-conferring mutations developed independently in each subtype, we suspect that the resistance-conferring mutations were passed to the different subtypes via recombination. Recombination is the primary means by which drug resistance is transferred in vivo between strains of HIV, so recombination at these locations involving drug resistant strains would allow successful transfer of the resistance-conferring mutations between types of HIV.

Figure 2 shows the spatial posterior estimate of the treatment effect on recombination. We see two areas of increased recombination, one from nucleotide positions 1020-1170 and another from nucleotide positions 1900-2200. As an interesting side note, we see a marked decrease in mutation and recombination in the V3 loop that determines sequence specificity.

## 3.2 Antisense gene therapy study

In the VIRxSYS antisense gene therapy study, we have two populations of wild type HIV *in vitro*. The samples consist of 19 Env sequences from a control HIV population that was allowed to evolve neutrally in cell culture, along with 48 Env sequences sampled from an HIV population evolving in cell cultures that were transfected with the VIRxSYS antisense vector [7]. The antisense gene therapy vector targets nucleotide positions 1325 - 2249. Unlike drug therapy treatments, whose effect can be nullified by just one or two well placed mutations, a relatively large number of mutations are required to escape the effects of antisense gene therapy. We again quantify the treatment effect of exposure to the antisense vector by calculating the posterior mean and 95% HPD interval of the

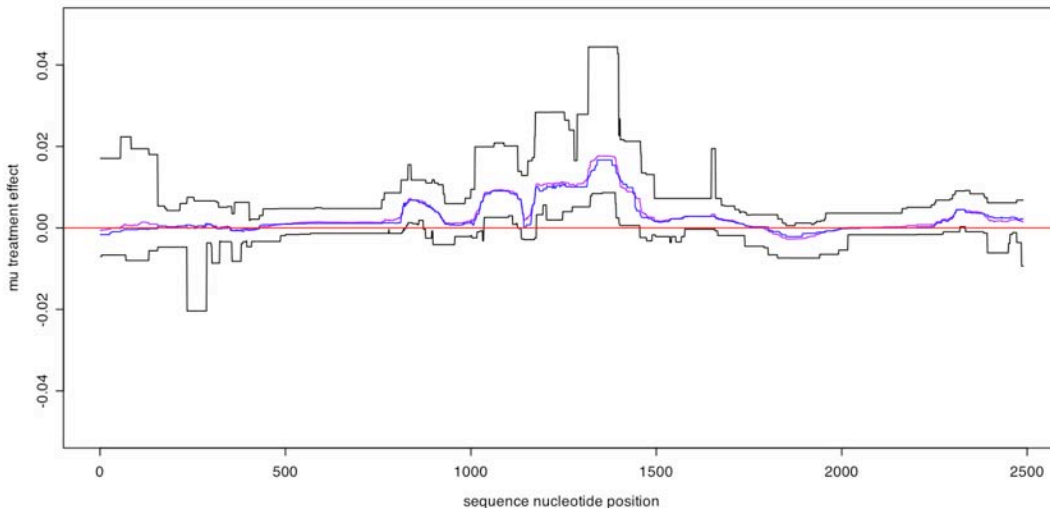

Figure 3: Posterior estimate of the effect of VIRxSYS antisense gene therapy on mutation rates. Blue line is posterior mean, Black lines are 95% HPD intervals.

difference in recombination rates $\rho^{\mathrm{T}} - \rho^{\mathrm{C}}$ and mutation rates $\mu^{\mathrm{T}} - \mu^{\mathrm{C}}$ at each position of the Env sequence.

Figures 3 and 4 show the posterior estimate of the treatment's effect on mutation and recombination, respectively. The most striking feature of the plots is the area of significantly elevated mutation in the treatment sequences. The leftmost region of the highest plateau corresponds to nucleotide position 1325, the 5' boundary of the antisense target region. This area of heightened mutation overlaps with the target region for around 425 nucleotides in the 3' direction. We see fewer differences in the recombination rate, suggesting that mutation is the primary mechanism of evolutionary response to the antisense vector. In fact, we estimate lower recombination rates in the target region of the treatment sequences relative to the control sequences.

## 4   Discussion

We have introduced a hierarchical model for the estimation of evolutionary escape response in a population exposed to therapeutic challenge. The escape response is quantified by mutation and recombination rate parameters. Our method allows for spatial heterogeneity in these mutation and recombination rates. It estimates differences between treatment and control sample parameters, with parameter values encouraged to be similar between the two populations except where the data suggests otherwise. We applied our procedure to sequence data from two different HIV therapy studies, detecting evolutionary responses in both studies that are of biological interest and may be relevant to the design of future HIV treatments.

Although virological problems motivated the creation of our model, it applies more generally to two-sample data sets of nucleic acid sequences drawn from any population. The model is particularly relevant for populations in which the recombination rate is a substantial fraction of the mutation rate, since simpler models which ignore recombination can produce seriously misleading results.

### Acknowledgements

This research was supported by a grant from the University of Pennsylvania Center for AIDS Research. Thanks to Neelanjana Ray, Jessamina Harrison, Robert Doms, Matthew Stephens and Gwen Binder for helpful discussions.

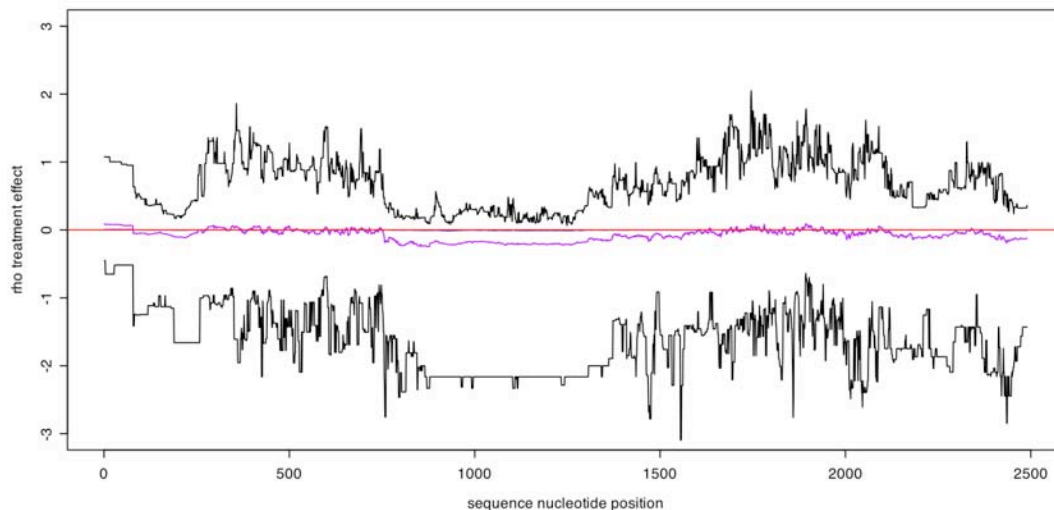

Figure 4: Posterior estimate of the effect of VIRxSYS antisense gene therapy on recombination rates. Blue line is posterior mean, Black lines are 95% HPD intervals.

## References

[1] P. J. Green. Reversible jump Markov chain Monte Carlo computation and Bayesian model determination. *Biometrika*, 82:711–731, 1995.

[2] R. C. Griffiths and P. Marjoram. An ancestral recombination graph. In *Progress in Population Genetics and Human Evolution*, pages 257–270. Springer Verlag, 1997.

[3] J. Hein, M. Schierup, and C. Wiuf. *Gene Genealogies, Variation and Evolution: A Primer in Coalescent Theory*. Oxford University Press, 2005.

[4] R. R. Hudson. Properties of a neutral allele model with intragenic recombination. *Theoretical Population Biology*, 23:183–201, 1983.

[5] J. F. C. Kingman. The coalescent. *Stochastic Processes and Their Applications*, 13:235–248, 1982.

[6] N. Li and M. Stephens. Modeling linkage disequilibrium and identifying recombination hotspots using single-nucleotide polymorphism data. *Genetics*, 165:2213–2233, December 2003.

[7] X. Lu, Q. Yu, G. Binder, Z. Chen, T. Slepushkina, J. Rossi, and B. Dropulic. Antisense-mediated inhibition of human immunodeficiency virus (HIV) replication by use of an HIV type 1-based vector results in severely attenuated mutants incapable of developing resistance. *Journal of Virology*, 78:7079–7088, 2004.

[8] G. McGuire, M. Denham, and D. Balding. Models of sequence evolution for DNA sequences containing gaps. *Molecular Biology and Evolution*, 18(4):481–490, 2001.

[9] N. Ray, J. Harrison, L. Blackburn, J. Martin, S. Deeks, and R. Doms. Clinical resistance to enfuvirtide does not affect susceptibility of human immunodeficiency virus type 1 to other classes of entry inhibitors. *Journal of Virology*, 81:3240–3250, 2007.

[10] M. H. Schierup and J. Hein. Consequences of recombination on traditional phylogenetic analysis. *Genetics*, 156:879–891, 2000.

[11] C. Wild, T. Greenwell, and T. Matthews. A synthetic peptide from HIV-1 gp41 is a potent inhibitor of virus mediated cell-cell fusion. *AIDS Research and Human Retroviruses*, 9:1051–1053, 1993.

[12] D. Wilson and G. McVean. Estimating diversifying selection and functional constraint in the presence of recombination. *Genetics*, 172:1411–1425, 2006.

